# An Auditory Localization and Coordinate Transform Chip

**Timothy K. Horiuchi**
timmer@cns.caltech.edu
Computation and Neural Systems Program
California Institute of Technology
Pasadena, CA 91125

## Abstract

The localization and orientation to various novel or interesting events in the environment is a critical sensorimotor ability in all animals, predator or prey. In mammals, the superior colliculus (SC) plays a major role in this behavior, the deeper layers exhibiting topographically mapped responses to visual, auditory, and somatosensory stimuli. Sensory information arriving from different modalities should then be represented in the same coordinate frame. Auditory cues, in particular, are thought to be computed in head-based coordinates which must then be transformed to retinal coordinates. In this paper, an analog VLSI implementation for auditory localization in the azimuthal plane is described which extends the architecture proposed for the barn owl to a primate eye movement system where further transformation is required. This transformation is intended to model the projection in primates from auditory cortical areas to the deeper layers of the primate superior colliculus. This system is interfaced with an analog VLSI-based saccadic eye movement system also being constructed in our laboratory.

## Introduction

Auditory localization has been studied in many animals, particularly the barn owl. Most birds have a resolution of only 10 to 20 degrees, but owls are able to orient

to sound with an accuracy of 1 to 2 degrees which is comparable with humans. One important cue for localizing sounds is the relative time of arrival of a sound to two spatially separated ears. A neural architecture first described by Jeffress (1948) for measuring this time difference has been shown to exist in the barn owl auditory localization system (Konishi 1986). An analog VLSI implementation of the barn owl system constructed by Lazzaro (1990) is extended here to include a transformation from head coordinates to retinal coordinates.

In comparison to the barn owl, the neurophysiology of auditory localization in cats and primates is not as well understood and a clear map of auditory space does not appear to be present in the inferior colliculus as it is in the owl. It has been suggested that cortical auditory regions may provide the head-based map of auditory space (Groh and Sparks 1992).

In primates, where much of the oculomotor system is based in retinotopic coordinates, head-based information must ultimately be transformed in order to be used. While other models of coordinate transformation have been proposed for visual information (e.g. Zipser and Andersen 1988, Krommenhoek *et al.* 1993) and for auditory information (Groh and Sparks 1992), the model of coordinate transformation used in this system is a switching network which shifts the entire projection of the head-based map of auditory space onto a retinotopic "colliculus" circuit. This particular model is similar to a basis function approach where intermediate units have compact receptive fields in an eye-position / head-based azimuth space and the output units sum the outputs of a subset of these units.

The auditory localization system described here provides acoustic target information to an analog VLSI-based saccadic eye movement system (Horiuchi, Bishofberger, and Koch 1994) being developed in our laboratory for multimodal operation.

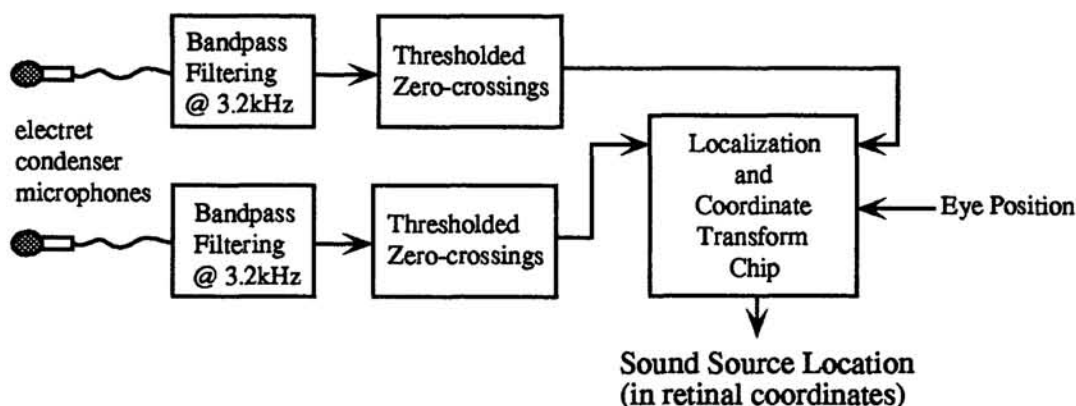

Figure 1: Block diagram of the auditory localization system. The analog front end consists of external discrete analog electronics.

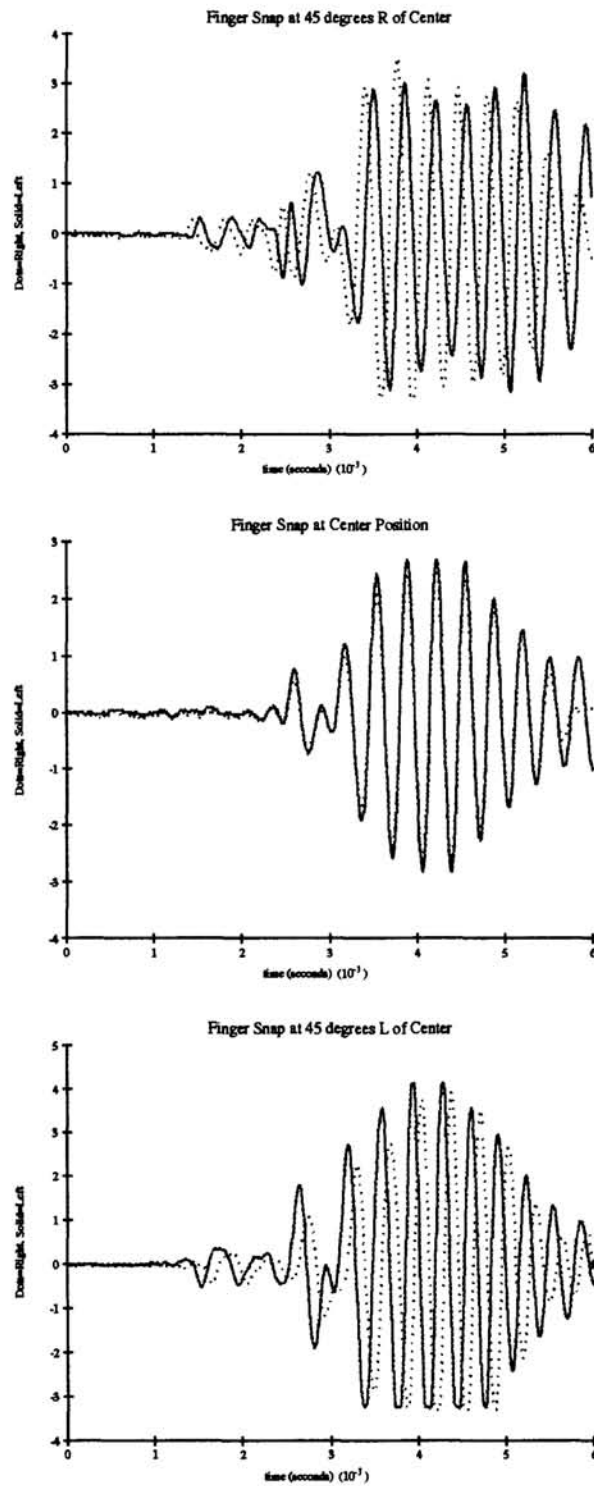

Figure 2: Filtered signals of the left and right microphones from 3 different angles.

## The Localization System

The analog front-end of the system (see figure 1) consists of three basic components, the microphones, the filter stage, and the thresholded, zero-crossing stage. Two microphones are placed with their centers about 2 inches apart. For any given time difference in arrival of acoustic stimuli, there are many possible locations from which the sound could have originated. These points describe a hyperbola with the two microphones as the two foci. If the sound source is distant enough, we can estimate the angle since the hyperbola approaches an asymptote. The current system operates on a single frequency and the inter-microphone distance has been chosen to be just under one wavelength apart at the filter frequency. The filter frequency chosen was 3.2 kHz because the author's finger snap, used extensively during development contained a large component at that frequency. The next step in the computation consists of triggering a digital pulse at the moment of zero-crossing if the acoustic signal is large enough.

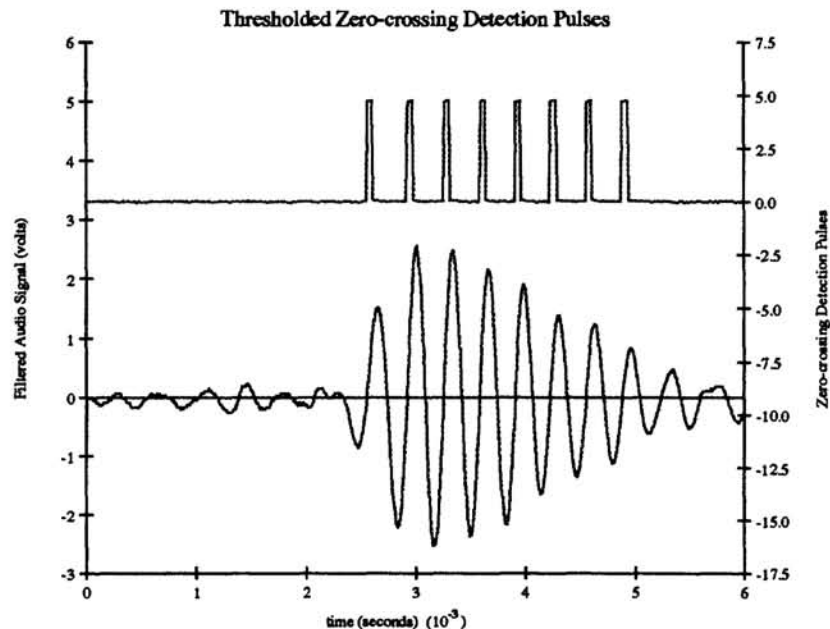

Figure 3: Example of output pulses from the external circuitry. Zero phase is chosen to be the positive-slope zero-crossing. Top: Digital pulses are generated at the time of zero phase for signals whose derivative is larger than a preset threshold. Bottom: 3.2 kHz Bandpass filtered signal for a finger snap.

## Phase Detection and Coordinate Transform in Analog VLSI

The analog VLSI component of the system consists of two axon delay lines (Mead 1988) which propagate the left and right microphone pulse signals in opposing directions in order to compute the cross correlation (see Fig 4.) The location of the peak in this correlation technique represents the relative phase of the two signals.

This technique is described in more detail and with more biological justification by Lazzaro (1990). The current implementation contains 15 axon circuits in each delay line. This is shown in figure 4. At each position in the correlation delay line are logical AND circuits which output a logic one when there are two active axon units at that location. Since these units only turn on for specific time delays, they define auditory "receptive fields". The output of this subsystem are 15 digital lines which are passed on to the coordinate transform.

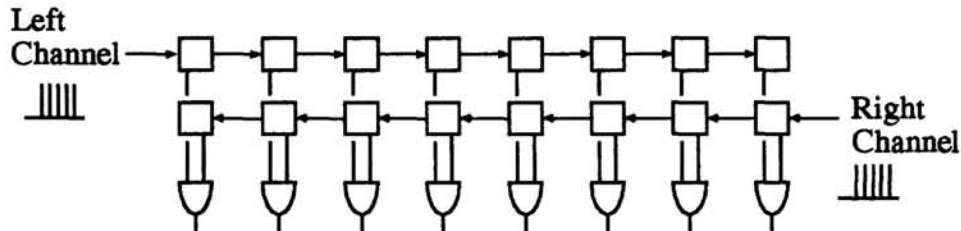

Figure 4: Diagram of the double axon delay line which accepts digital spikes on the inputs and propagates them across the array. Whenever two spikes meet, a pulse is generated on the output AND units. The position of the AND circuit which gets activated indicates the relative time of arrival of the left and right inputs. NOTE: the actual circuit contains 15 axon units.

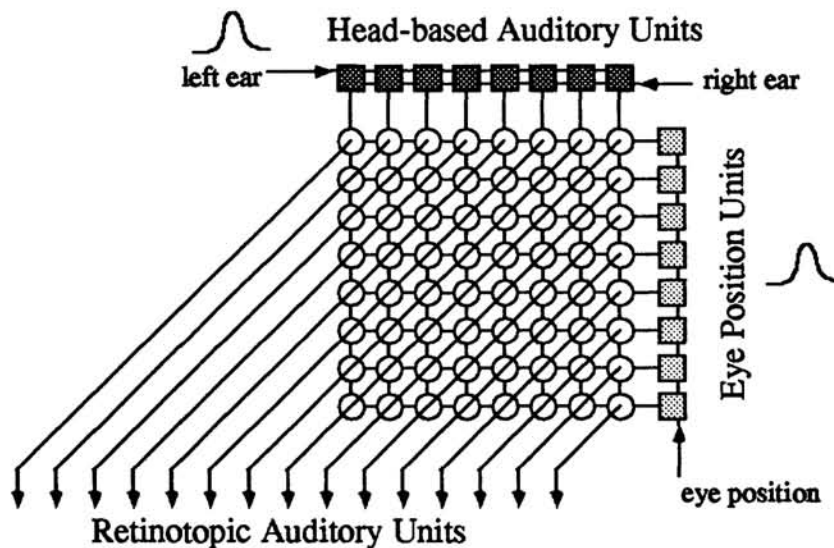

Figure 5:

For the one-dimensional case described in this project, the appropriate transform from head to retinal coordinates is a rotation which subtracts the eye position. The eye position information on the chip is represented as a voltage which activates one of the eye position units. The spatial pattern of activation from the auditory units is then "steered" to the output stage with the appropriate shift. (See figure 5). This

is similar to a shift scheme proposed by Pitts and McCulloch (1947) for obtaining pitch invariance for chord recognition. The eye position units are constructed from an array of "bump" circuits (Delbrück 1993) which compare the eye position voltage with its local voltage reference. The two dimensional array of intermediate units take the digital signal from the auditory units and switch the "bump" currents onto the output lines. The output current lines drive the inputs of a centroid circuit.

The current implementation of the shift can be viewed as a basis function approach where a population of intermediate units respond to limited "ball-like" regions in the two-dimensional space of horizontal eye position and sound source azimuth (head-coordinates). The output units then sum the outputs of only those intermediate units which represent the same retinal location. It should be noted that this coordinate transformation is closely related to the "dendrite model" proposed for the projection of cortical auditory information to the deep SC by Groh and Sparks (1992).

The final output stage converts this spatial array of current carrying lines into a single output voltage which represents the centroid of the stimulus in retinal coordinates. This centroid circuit (DeWeerth 1991) is intended to represent the primate SC where a similar computation is believed to occur.

## Results and Conclusions

Figure 6 shows three plots of the chip's output voltage as a function of the inter-pulse time interval. Figure 7 shows three plots of the full system's output voltage for different eye position voltages. The output is roughly linear with azimuth and linear with eye position voltage. In operation, the system input consists of a sound entering the two microphones and the output consists of an analog voltage representing the position of the sound source and a digital signal indicating that the analog data is valid.

The auditory localization system described here is currently in use with an analog VLSI-based model of the primate saccadic system to expand its operation into the auditory domain (Horiuchi, Bishofberger, & Koch 1994). In addition to the effort of our laboratory to model and understand biological computing structures in real-time systems, we are exploring the use of these low power integrated sensors in portable applications such as mobile robotics. Analog VLSI provides a compact and efficient implementation for many neuromorphic computing architectures which can potentially be used to provide, small, fast, low power sensors for a wide variety of applications.

## Acknowledgements

The author would like to acknowledge Prof. Christof Koch for his academic support and use of laboratory facilities for this project, Brooks Bishofberger for his assistance in constructing some of the discrete electronics and Prof. Carver Mead for running the CNS184 course under which this chip was fabricated.
The author is supported by an AASERT grant from the Office of Naval Research.

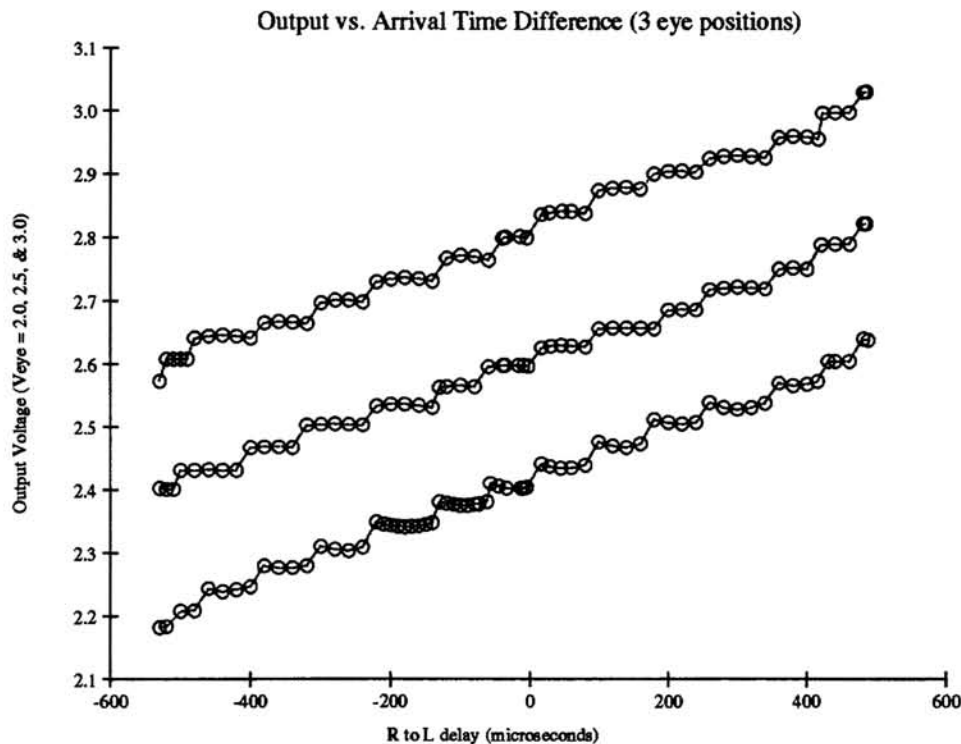

Figure 6: Chip output vs. input pulse timing: The chip was driven with a signal generator and the output voltage was plotted for three different eye position voltages. Due to the discretized nature of the axon, there are only 15 axon locations at which pulses can meet. This creates the staircase response.

# References

T. Delbrück (1993) Investigations of Analog VLSI Visual Transduction and Motion Processing, Ph.D. Thesis, California Institute of Technology

J. Groh and D. Sparks (1992) 2 Models for Transforming Auditory Signals from Head-Centered to Eye-Centered Coordinates *Biol. Cybern.* 67(4) 291-302.

T. Horiuchi, B. Bishofberger, & C. Koch, (1994) An Analog VLSI-based Saccadic System, In (ed.), *Advances in Neural Information Processing Systems 6* San Mateo, CA: Morgan Kaufman

L. A. Jeffress (1948) A Place Theory of Sound Localization *J. Comp. Physiol. Psychol.* 41: 35-39.

M. Konishi (1986) Centrally Synthesized Maps of Sensory Space. *TINS* April, pp. 163-168.

K. P. Krommenhoek, A. J. Van Opstal, C. C. A. M. Gielen, J. A. ,M. Van Gisbergen. (1993) Remapping of Neural Activity in the Motor Colliculus: A Neural Network Study. *Vision Research* 33(9):1287-1298.

J. Lazzaro. (1990) Silicon Models of Early Audition, Ph.D. Thesis, California Institute of Technology

C. Mead, (1988) *Analog VLSI and Neural Systems* Menlo Park: Addison-Wesley

W. Pitts and W. S. McCulloch, (1947) How we know universals: the perception of auditory and visual forms. *Bulletin of Mathematical Biophysics* 9:127-147.

D. Zipser and R. A. Andersen (1988) A back-propagation programmed network that simulates response properties of a subset of posterior parietal neurons. *Nature* 331:679-684.

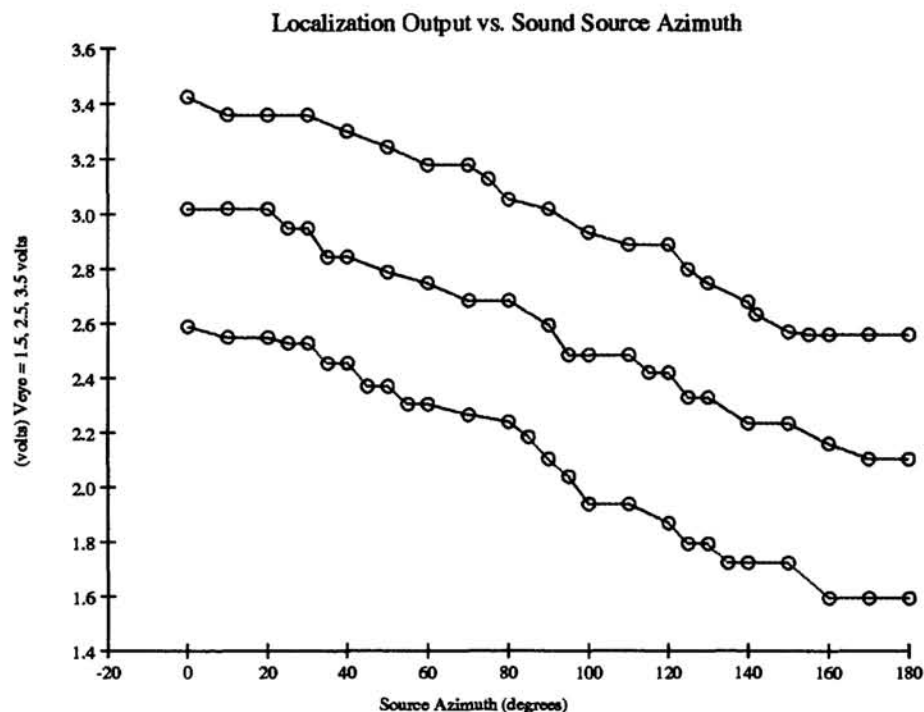

Figure 7: Performance of the full system on continuous input (sinusoidal) delivered by a speaker from different angles. Note that 90 degrees denotes the center position. The three plots are the outputs for three different settings of the eye position input voltage.